# Learning Motion Style Synthesis
# from Perceptual Observations

**Lorenzo Torresani**
Riya, Inc.
lorenzo@riya.com

**Peggy Hackney**
Integrated Movement Studies
pjhackney@aol.com

**Christoph Bregler**
New York University
chris.bregler@nyu.edu

## Abstract

This paper presents an algorithm for synthesis of human motion in specified styles. We use a theory of movement observation (Laban Movement Analysis) to describe movement styles as points in a multi-dimensional perceptual space. We cast the task of learning to synthesize desired movement styles as a regression problem: sequences generated via space-time interpolation of motion capture data are used to learn a nonlinear mapping between animation parameters and movement styles in perceptual space. We demonstrate that the learned model can apply a variety of motion styles to pre-recorded motion sequences and it can extrapolate styles not originally included in the training data.

## 1 Introduction

Human motion perception can be generally thought of as the result of interaction of two factors, traditionally termed content and style. Content generally refers to the nature of the action in the movement (e.g. walking, reaching, etc.), while style denotes the particular way that action is performed. In computer animation, the separation of the underlying content of a movement from its stylistic characteristics is particularly important. For example, a system that can synthesize stylistic variations of a given action would be a useful tool for animators. In this work we address such a problem by proposing a system that applies user-specified styles to motion sequences. Specifically, given as input a target motion style and an arbitrary animation or pre-recorded motion, we want to synthesize a novel sequence that preserves the content of the original input motion but exhibits style similar to the user-specified target.

Our approach is inspired by two classes of methods that have successfully emerged within the genre of data-driven animation: sample-based concatenation methods, and techniques based on learned parametric models. Concatenative synthesis techniques [15, 1, 11] are based on the simple idea of generating novel movements by concatenation of motion capture snippets. Since motion is produced by cutting and pasting pre-recorded examples, the resulting animations achieve realism similar to that of pure motion-capture play back. Snippet concatenation can produce novel content by generating arbitrarily complex new movements. However, this approach is restricted to synthesize only the subset of styles originally contained in the input database. Sample-based concatenation techniques are unable to produce novel stylistic variations and cannot generalize style differences from the existing examples. In recent years, several machine learning animation systems [2, 12, 9] have been proposed that attempt to overcome some of these limitations. Unfortunately, most of these methods learn simple parametric motion models that are unable to fully capture the subtleties and complexities of human movement. As a consequence, animations resulting from these systems are often plagued by low quality and scarce realism.

The technique introduced in this paper is a compromise between the pure concatenative approaches and the methods based on learned parametric models. The aim is to maintain the animated precision of motion capture data, while introducing the flexibility of style changes achievable by learned

parametric models. Our system builds on the observation that stylistically novel, yet highly realistic animations can be generated via space-time interpolation of pairs of motion sequences. We propose to learn not a parametric function of the motion, but rather a parametric function of how the interpolation or extrapolation weights applied to data snippets relate to the styles of the output sequences. This allows us to create motions with arbitrary styles without compromising animation quality.

Several researchers have previously proposed the use of motion interpolation for synthesis of novel movement [18, 6, 10]. These approaches are based on the naïve assumption that motion interpolation produces styles corresponding precisely to the interpolation of the styles of the original sequences. In this paper we experimentally demonstrate that styles generated through motion interpolation are a rather complex function of styles and contents of the original snippets. We propose to explicitly learn the mapping between motion blending parameters and resulting animation styles. This enables our animation system not only to generate arbitrary stylistic variations of a given action, but, more importantly, to synthesize sequences matching user-specified stylistic characteristics. Our approach bears similarities with the Verbs and Adverbs work of Rose et al. [16], in which interpolation models parameterized by style attributes are learned for several actions, such as walking or reaching. Unlike this previously proposed algorithm, our solution can automatically identify sequences having similar content, and therefore does not require manual categorization of motions into classes of actions. This feature allows our algorithm to be used for style editing of sequences without content specification by the user. Additionally, while the Verb and Adverb system characterizes motion styles in terms of difficult-to-measure emotional attributes, such as sad or clueless, our approach relies on a theory of movement observation, Laban Movement Analysis, describing styles by means of a set of rigorously defined perceptual attributes.

## 2  The LMA Framework

In computer animation literature motion style is a vaguely defined concept. In our work, we describe motion styles according to a movement notation system, called Laban Movement Analysis or LMA [7]. We focus on a subset of Laban Movement Analysis: the "LMA-Effort" dimensions. This system does not attempt to describe the coarse aspects of a motion, e.g. whether someone is walking, or swinging his/her arm. Instead, it targets the subtle differences in motion style, e.g. is the movement "bound" or "free"? Each LMA-Effort factor varies in intensity between opposing poles, and takes values in a continuous range. The factors are briefly described as follows:

1. The "LMA-Effort Factor of Flow" defines the continuity of the movement. The two opposing poles are "Free" (fluid, released), and "Bound" (controlled, contained, restrained).
2. The "LMA-Effort Factor of Weight" is about the relationship of the movement to gravity. The two opposing extremes are "Light" (gentle, delicate, fine touch) and "Strong" (powerful, forceful, firm touch).
3. The "LMA-Effort Factor of Time" has to do with the persons inner attitude toward the time available, but not with how long it takes to perform the movement. The two opposing poles are "Sudden" (urgent, quick) and "Sustained" (stretching the time, indulging).
4. The "LMA-Effort Factor of Space" describes the directness of the movement. Generally, additional features not present in motion capture data, such as eye gaze, are necessary to detect this factor.

We use only the LMA-Effort factors of Flow, Weight, and Time. We model styles as points in a three-dimensional perceptual space derived by translating the LMA-Effort notations for each of these factors into numerical values ranging in the interval $[-3, 3]$.

## 3  Overview of the system

The key-idea of our work is to learn motion style synthesis from a training set of computer-generated animations. The training animations are observed by a human expert who assigns LMA labels to each sequence. This set of supervised data is used to learn a mapping between the space of motion styles and the animation system parameters. We next provide a high-level description of our system, while the following sections give specific details of each component.

## 3.1 Training: Learning the Style of Motion Interpolation

In order to train our system to synthesize motion styles, we employ a corpus of human motion sequences recorded with a motion capture system. We represent the motion as a time-varying vector of joint angles. In the training stage each motion sequence is manually segmented by an LMA human expert into fragments corresponding to fundamental actions or units of motions. Let $\mathbf{X}_i$ denote the joint angle data of the $i$-th fragment in the database.

**Step 1: Matching motion content.** We apply a motion matching algorithm to identify fragment pairs $(\mathbf{X}_i, \mathbf{X}_j)$ containing similar actions. Our motion matching algorithm is based on dynamic-time warping. This allows us to compare kinematic contents while factoring out differences in timing or acceleration, more often associated to variations in style.

**Step 2: Space-time interpolation.** We use these motion matches to augment the database with new synthetically-generated styles: given matching motion fragments $\mathbf{X}_i$, $\mathbf{X}_j$, and an interpolation parameter $\alpha$, space-time interpolation smoothly blends the kinematics and dynamics of the two fragments to produce a new motion $\mathbf{X}_{i,j}^{\alpha}$ with novel distinct style and timing.

**Step 3: Style interpolation learning.** Both the synthesized animations $\mathbf{X}_{i,j}^{\alpha}$ as well as the "seed" motion capture data $\mathbf{X}_i$ are labeled with LMA-Effort values by an LMA expert. Let $\mathbf{e}_i$ and $\mathbf{e}_{i,j}^{\alpha}$ denote the three-dimensional vectors encoding the LMA-Effort qualities of $\mathbf{X}_i$ and $\mathbf{X}_{i,j}^{\alpha}$, respectively. A non-linear regression model [5] is fitted to the LMA labels and the parameters $\alpha$ of the space-time interpolation algorithm. This regression defines a function $f$ predicting LMA-Effort factors $\mathbf{e}_{i,j}^{\alpha}$ from the style attributes and joint angle data of fragments $i$ and $j$:

$$\mathbf{e}_{i,j}^{\alpha} \quad = \quad f(\mathbf{X}_i, \mathbf{X}_j, \mathbf{e}_i, \mathbf{e}_j, \alpha) \tag{1}$$

This function-fitting stage allows us to learn how the knobs of our animation system relate to the perceptual space of movement styles.

## 3.2 Testing: Style Transfer

At testing stage we are given a motion sequence $\mathbf{Y}$, and a user-specified motion style $\bar{\mathbf{e}}$. The goal is to apply style $\bar{\mathbf{e}}$ to the input sequence $\mathbf{Y}$, without modifying the content of the motion. First, we use dynamic-time warping to segment the input sequence into snippets $\mathbf{Y}_i$, such that each snippet matches the content of a set of analogous motions $\{\mathbf{X}_{i_1}, ..., \mathbf{X}_{i_K}\}$ in the database. Among all possible pairwise blends $\mathbf{X}_{i_k, i_l}^{\alpha}$ of examples in the set $\{\mathbf{X}_{i_1}, ..., \mathbf{X}_{i_K}\}$, we determine the one that provides the best approximation to the target style $\bar{\mathbf{e}}$. This objective can be formulated as

$$\alpha^*, k^*, l^* \leftarrow \underset{\alpha, k, l}{\arg\min} \, ||\bar{\mathbf{e}} - f(\mathbf{X}_{i_k}, \mathbf{X}_{i_l}, \mathbf{e}_{i_k}, \mathbf{e}_{i_l}, \alpha)|| \tag{2}$$

The animation resulting from space-time interpolation of fragments $\mathbf{X}_{i_{k^*}}$ and $\mathbf{X}_{i_{l^*}}$ with parameter $\alpha^*$ will exhibit content similar to that of snippet $\mathbf{Y}_i$ and style approximating the target $\bar{\mathbf{e}}$. Concatenating these artificially-generated snippets will produce the desired output.

## 4 Matching motion content

The objective of the matching algorithm is to identify pairs of sequences having similar motion content or consisting of analogous activities. The method should ignore variations in the style with which movements are performed. Previous work [2, 12] has shown that the differences in movement styles can be found by examining the parameters of timing and movement acceleration. By contrast, an action is primarily characterized by changes of body configurations in space rather than over time. Thus we compare the content of two motions by identifying similar spatial body poses while allowing for potentially large differences in timing. Specifically, we define the content similarity between motion snippets $\mathbf{X}_i$ and $\mathbf{X}_j$, as the minimum sum of their squared joint angle differences $SSD(\mathbf{X}_i, \mathbf{X}_i)$ under a dynamic time warping path. Let $d(p, q) = ||\mathbf{X}_i(p) - \mathbf{X}_j(q)||^2$ be our local measure of the distance between spatial body configurations $\mathbf{X}_i$ at frame $p$ and $\mathbf{X}_j$ at frame $q$. Let $T_i$

be the number of frames in sequence $i$ and $L$ the variable length of a time path $\mathbf{w}(n) = (\mathbf{p}(n), \mathbf{q}(n))$ aligning the two snippets. We can then formally define $SSD(\mathbf{X}_i, \mathbf{X}_i)$ as:

$$SSD(\mathbf{X}_i, \mathbf{X}_i) = \min_{\mathbf{w}} \sum_n d(\mathbf{w}(n)) \tag{3}$$

subject to constraints:

$$\mathbf{p}(1) = 1, \mathbf{q}(1) = 1, \mathbf{p}(L) = T_i, \mathbf{q}(L) = T_j \tag{4}$$

$$\text{if } \mathbf{w}(n) = (p, q) \text{ then } \mathbf{w}(n-1) \in \{(p-1, q), (p-1, q-1), (p, q-1)\} \tag{5}$$

We say that two motions $i$ and $j$ have similar content if $SSD(\mathbf{X}_i, \mathbf{X}_i)$ is below a certain value.

## 5 Space-time interpolation

A time warping strategy is also employed to synthesize novel animations from the pairs of content-matching examples found by the algorithm outlined in the previous section. Given matching snippets $\mathbf{X}_i$ and $\mathbf{X}_j$, the objective is to generate a stylistically novel sequence that maintains the content of the two original motions. The idea is to induce changes in style by acting primarily on the timings of the motions. Let $\mathbf{w}^* = (\mathbf{p}^*, \mathbf{q}^*)$ be the path minimizing Equation 3. This path defines a time alignment between the two sequences. We can interpret frame correspondences $(\mathbf{p}^*(n), \mathbf{q}^*(n))$ for $n = 1, ..., L$, as discrete samples from a continuous 2D curve parameterized by $n$. Resampling $\mathbf{X}_i$ and $\mathbf{X}_j$ along this curve will produce synchronized versions of the two animations, but with new timings. Suppose parameter values $n_1^0, ...., n_{T_i}^0$ are chosen such that $\mathbf{p}^*(n_k^0) = k$. Then $\mathbf{X}_i(\mathbf{p}^*(n_k^0))$ will be replayed with its original timing. However, if we use these same parameter values on sequence $\mathbf{X}_j$ (i.e. we estimate joint angles $\mathbf{X}_j$ at time steps $\mathbf{q}^*(n_k^0)$) then the resampled motion will correspond to playing sequence $j$ with the timing of sequence $i$. Similarly, $n_1^1, ...., n_{T_j}^1$ can be chosen, such that $\mathbf{q}^*(n_k^1) = k$, and these parameter values can be used to synthesize motion $i$ with the timing of motion $j$. It is also possible to smoothly interpolate between these two scenarios according to an interpolation parameter $\alpha \in [0, 1]$ to produce intermediate time warps. This will result in a time path of length $T_{ij}^\alpha = (1-\alpha)T_i + \alpha T_j$. Let us indicate with $n_1^\alpha, ...., n_{T_{ij}^\alpha}^\alpha$ the path parameter values obtained from this time interpolation. New stylistic versions of motions $i$ and $j$ can be produced by estimating the joint angles $\mathbf{X}_i$ and $\mathbf{X}_j$ at $\mathbf{p}^*(n_k^\alpha)$ and $\mathbf{q}^*(n_k^\alpha)$, respectively. The two resulting sequences will move in synchrony according to the new intermediate timing. From these two synchronized sequences, a novel motion $\mathbf{X}_{i,j}^\alpha$ can be generated by averaging the joint angles according to mixing coefficients $(1-\alpha)$ and $\alpha$: $\mathbf{X}_{i,j}^\alpha(k) = (1-\alpha)\mathbf{X}_i(\mathbf{p}^*(n_k^\alpha)) + \alpha\mathbf{X}_j(\mathbf{q}^*(n_k^\alpha))$. The synthesized motion $\mathbf{X}_{i,j}^\alpha$ will display content similar to that of $\mathbf{X}_i$ and $\mathbf{X}_j$, but it will have distinct style. We call this procedure "space-time interpolation", as it modifies the spatial body configurations and the timings of sequences.

## 6 Learning style interpolation

Given a pair of content-matching snippets $\mathbf{X}_i$ and $\mathbf{X}_j$, our goal is to determine the parameter $\alpha$ that needs to be applied to space-time interpolation in order to produce a motion $\mathbf{X}_{i,j}^\alpha$ exhibiting target style $\bar{\mathbf{e}}$. We propose to solve this task by learning to predict the LMA-Effort qualities of animations synthesized by space-time interpolation. The training data for this supervised learning task consists of our seed motion sequences $\{\mathbf{X}_i\}$ in the database, a set of interpolated motions $\{\mathbf{X}_{i,j}^\alpha\}$, and the corresponding LMA-Effort qualities $\{\mathbf{e}_i\}$, $\{\mathbf{e}_{i,j}^\alpha\}$ observed by an LMA human expert. In order to maintain a consistent data size, we stretch or shrink the time trajectories of the joint angles $\{\mathbf{X}_i\}$ to a set length. In order to avoid overfitting, we compress further the motion data by projecting it onto a low-dimensional linear subspace computed using Principal Component Analysis (PCA). In many of the test cases, we found it was sufficient to retain only the first two or three principal components in order to obtain a discriminative representation of the motion contents. Let $\mathbf{c}_i$ denote the vector containing the PCA coefficients computed from $\mathbf{X}_i$. Let $\mathbf{z}_{i,j}^\alpha = [\mathbf{c}_i^T, \mathbf{c}_j^T, \mathbf{e}_i^T, \mathbf{e}_j^T, \alpha]^T$. We pose the task of predicting LMA-Effort qualities as a function approximation problem: the goal is to learn the optimal parameters $\theta$ of a parameterized function $f(\mathbf{z}_{i,j}^\alpha, \theta)$ that models the dependencies

between $\mathbf{z}_{i,j}^{\alpha}$ and the observed LMA-Effort values $\mathbf{e}_{i,j}^{\alpha}$. Parameters $\theta$ are chosen so as to minimize the objective function:

$$E(\theta) \quad = \quad U \sum L(f(\mathbf{z}_{i,j}^{\alpha}, \theta) - \mathbf{e}_{i,j}^{\alpha}) + ||\theta||^2 \tag{6}$$

where $L$ is a general loss function and $U$ is a regularization constant aimed at avoiding overfitting and improving generalization. We experimented with several function parameterizations and loss functions applied to our problem. The simplest of the adopted approaches is linear ridge regression [4], which corresponds to choosing the loss function $L$ to be quadratic (i.e. $L(.) = (.)^2$) and $f$ to be linear in input space:

$$f(\mathbf{z}, \theta) \quad = \quad \mathbf{z}^T \cdot \theta \tag{7}$$

We also applied kernel ridge regression, resulting from mapping the input vectors $\mathbf{z}$ into features of a higher-dimensional space via a nonlinear function $\Phi\colon \mathbf{z} \to \Phi(\mathbf{z})$. In order to avoid the explicit computation of the vectors $\Phi(\mathbf{z}_j)$ in the high-dimensional feature space, we apply the kernel trick and choose mappings $\Phi$ such that the inner product $\Phi(\mathbf{z}_i)^T \cdot \Phi(\mathbf{z}_j)$ can be computed via a kernel function $k(\mathbf{z}_i, \mathbf{z}_j)$ of the inputs. We compared the performance of kernel ridge regression with that of support vector regression [5]. While kernel ridge regression requires us to store all training examples in order to evaluate function $f$ at a given input, support vector regression overcomes this limitation by using an $\epsilon$-insensitive loss function [17]. The resulting $f$ can be evaluated using only a subset of the training data, the set of support vectors.

# 7 Testing: Style Transfer

We can restate our initial objective as follows: given an input motion sequence $\mathbf{Y}$ in unknown style, and a target motion style $\bar{\mathbf{e}}$ specified by LMA-Effort values, we want to synthesize a sequence having style $\bar{\mathbf{e}}$ and content analogous to that of motion $\mathbf{Y}$. A naïve approach to this problem is to seek in the motion database a pair of sequences having content similar to $\mathbf{Y}$ and whose interpolation can approximate style $\bar{\mathbf{e}}$. The learned function $f$ can be used to determine the pair of motions and the interpolation parameter $\alpha$ that produce the best approximation to $\bar{\mathbf{e}}$. However, such an approach is destined to fail as $\mathbf{Y}$ can be any arbitrarily long and complex sequence, possibly consisting of several movements performed one after the other. As a consequence, we might not have in the database examples that match sequence $\mathbf{Y}$ in its entirety.

## 7.1 Input segmentation and matching

The solution that we propose is inspired by concatenative methods. The idea is to determine the concatenation of database motion examples $[\mathbf{X}_1, ..., \mathbf{X}_N]$ that best matches the content of input sequence $\mathbf{Y}$. Our approach relies again on dynamic programming and can be interpreted as a generalization of the dynamic time warping technique presented in Section 4, for the case when a time alignment is sought between a given sequence and a concatenation of a variable number of examples chosen from a set. Let $d(p, q, i)$ be the sum of squared differences between the joint angles of sequence $\mathbf{Y}$ at frame $p$ and those of example $\mathbf{X}_i$ at frame $q$. The goal is to recover the time warping path $\mathbf{w}(n) = (\mathbf{p}(n), \mathbf{q}(n), \mathbf{i}(n))$ that minimizes the global error

$$\min_{\mathbf{w}} \sum_n d(\mathbf{w}(n)) \tag{8}$$

subject to basic segment transition and endpoint constraints. Transitions constraints are enforced to guarantee that time order is preserved and that no time frames are omitted. Endpoint constraints require that the time path starts at beginning frames and finishes at ending frames of the sequences. The above mentioned conditions can be formalized as follows:

$$\text{if } \mathbf{w}(n) = (p, 1, i), \text{ then } \mathbf{w}(n-1) \in \{(p-1, 1, i), (p-1, T_j, j)\text{for } j = 1, ..., J\} \tag{9}$$

$$\text{if } \mathbf{w}(n) = (p, q, i) \text{ and } q > 1, \text{ then } \mathbf{w}(n-1) \in \{(p-1, q, i), (p-1, q-1, i), (p, q-1, i)\} \tag{10}$$

$$\mathbf{p}(1) = 1, \mathbf{q}(1) = 1, \mathbf{p}(L) = T, \mathbf{q}(L) = T_{\mathbf{i}(L)} \tag{11}$$

where $J$ denotes the number of fragments in the database, $L$ the length of the time warping path, $T$ the number of frames of the input sequence, and $T_j$ the length of the $j$-th fragment in the database.

Table 1: Mean squared error on LMA-Effort prediction for different function approximation methods

| Function Approxim. Method | Linear Interpolation | Linear Ridge Regression | Kernel Ridge Regression | Support Vector Regression |
|---|---|---|---|---|
| Flow MSE | 0.65 | 1.03 | 0.50 | 0.48 |
| Weight MSE | 0.97 | 1.04 | 0.39 | 0.48 |
| Time MSE | 1.01 | 1.01 | 0.60 | 0.61 |

The global minimum of the objective in Equation (8), subject to constraints (9),(10),(11), can be found using a dynamic programming method originally developed by Ney [14] for the problem of connected word recognition in speech data. Note that this approach induces a segmentation of the input sequence $\mathbf{Y}$ into snippets $[\mathbf{Y}_1, ..., \mathbf{Y}_N]$, matching the examples in the optimal concatenation $[\mathbf{X}_1, ..., \mathbf{X}_N]$.

### 7.2 Piecewise Style synthesis

The final step of our algorithm uses the concatenation of examples $[\mathbf{X}_1, ..., \mathbf{X}_N]$ determined by the method outlined in the previous section to synthesize a version of motion $\mathbf{Y}$ in style $\bar{\mathbf{e}}$. For each $\mathbf{X}_i$ in $[\mathbf{X}_1, ..., \mathbf{X}_N]$, we identify the $K$ most similar database examples according to the criterion defined in Equation 3. Let $\{\mathbf{X}_{i_1}, ..., \mathbf{X}_{i_K}\}$ denote the $K$ content-neighbors of $\mathbf{X}_i$ and $\{\mathbf{e}_{i_1}, ..., \mathbf{e}_{i_K}\}$ their LMA-Effort values. $\{\mathbf{X}_{i_1}, ..., \mathbf{X}_{i_K}\}$ defines a cluster of examples having content similar to that of snippet $\mathbf{Y}_i$. The final goal then is to replace each snippet $\mathbf{Y}_i$ with a pairwise blend of examples in its cluster so as to produce a motion exhibiting style $\bar{\mathbf{e}}$. Formally, this is achieved by determining the pair of examples $(i_{k^*}, i_{l^*})$ in $\mathbf{Y}_i$'s cluster, and the interpolation weight $\alpha^*$ that provide the best approximation to target style $\bar{\mathbf{e}}$, according to the learned style-prediction function $f$:

$$\alpha^*, k^*, l^* \leftarrow \arg\min_{\alpha,k,l} ||\bar{\mathbf{e}} - f(\mathbf{z}^{\alpha}_{i_k, i_l})|| \tag{12}$$

Minimization of this objective is achieved by first finding the optimal $\alpha$ for each possible pair $(i_k, i_l)$ of candidate motion fragments. We then select the pair $(i_{k^*}, i_{l^*})$ providing the minimum deviation from the target style $\bar{\mathbf{e}}$. In order to estimate the optimal values of $\alpha$ for pair $(i_k, i_l)$, we evaluate $f(\mathbf{z}^{\alpha}_{i_k, i_l})$ for $M$ values of $\alpha$ uniformly sampled in the interval [-0.25,1.25], and choose the value with the closest fit to the target style. We found that $f$ tends to vary smoothly as a function of $\alpha$, and thus a good estimate of the global minimum in the specified interval can be obtained even with a modest number $M$ of samples. The approximation is further refined using a golden section search [8] around the initial estimate. Note that, by allowing values $\alpha$ to be chosen in the range [-0.25,1.25] rather than [0,1], we give the algorithm the ability to extrapolate from existing motion styles.

Given optimal parameters $(\alpha^*, k^*, l^*)$, space-time interpolation of fragments $\mathbf{X}_{i_{k^*}}$ and $\mathbf{X}_{i_{l^*}}$ with parameter value $\alpha^*$ produces an animation with content similar to that of $\mathbf{Y}_i$ and style approximating the desired target $\bar{\mathbf{e}}$. This procedure is repeated for all snippets of $\mathbf{Y}$. The final animation is obtained by concatenating all of the fragments generated via interpolation with optimal parameters.

## 8 Experiments

The system was tested using a motion database consisting of 12 sequences performed by different professional dancers. The subjects were asked to perform a specific movement phrase in their own natural style. Each of the 12 sequences was segmented by an LMA expert into 5 fragments corresponding to the main actions in the phrase. All fragments were then automatically clustered into 5 content groups using the $SSD$ criterion outlined in section 4. The motions were recorded using a marker-based motion capture system. In order to derive joint angles, the 3D trajectories of the markers were fitted to a kinematic chain with 17 joints. The joint angles were represented with exponential maps [13], which have the property of being locally linear and thus particularly suitable for motion interpolation. From these 60 motion fragments, 105 novel motions were synthesized with space-time interpolation using random values of $\alpha$ in the range $[-0.25, 1.25]$. All motions, both those recorded and those artificially generated, were annotated with LMA-Effort qualities by

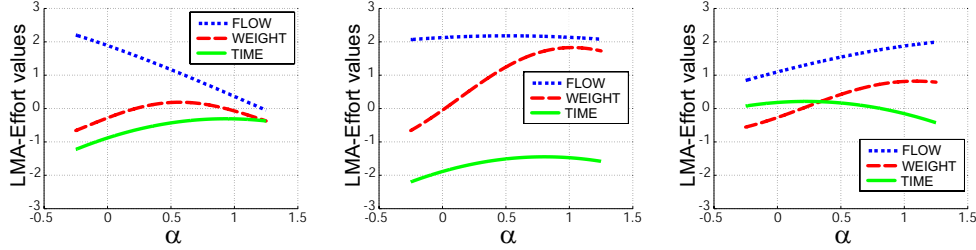

Figure 1: Sample LMA-Effort attributes estimated by kernel ridge regression on three different pairs of motions $(\mathbf{X}_i, \mathbf{X}_j)$ and for $\alpha$ varying in [-0.25, 1.25]. The Flow attribute appears to be almost linearly dependent on $\alpha$. By contrast, Weight and Time exhibit non-linear relations with the interpolation parameter.

an LMA expert. From this set of motions, 85 training examples were randomly selected to train the style regression models. The remaining 20 examples were used for testing.

Table 1 summarizes the LMA-Effort prediction performance in terms of mean squared error for the different function approximation models discussed in the paper. Results are reported by averaging over 500 runs of random splitting of the examples into training and testing sets. We include in our analysis the linear style interpolation model, commonly used in previous work. This model assumes that the style of a sequence generated via motion interpolation is equal to the interpolation of the styles of the two seed motions: $\mathbf{e}_{i,j}^{\alpha} = \alpha \mathbf{e}_i + (1 - \alpha)\mathbf{e}_j$. In all experiments involving kernel-based approximation methods, we used a Gaussian RBF kernel. The hyperparameters (i.e. the kernel and the regularization parameters) were tuned using tenfold cross-validation. Since the size of the training data is not overly large, it was possible to run kernel ridge regression without problems despite the absence of sparsity of this solution. The simple linear interpolation model performed reasonably well only on the Flow dimension. Overall, non-linear regression models proved to be much superior to the linear interpolation function, indicating that the style of sequences generates via space-time interpolation is a complex function of the original styles and motions.

Figure 1 shows the LMA-Effort qualities predicted by kernel ridge regression while varying $\alpha$ for three different sample values of the inputs $(\mathbf{X}_i, \mathbf{X}_j, \mathbf{e}_i, \mathbf{e}_j)$. Note that the shapes of the sample curves learned by kernel ridge regression for the Flow attribute suggest an almost linear dependence of Flow on $\alpha$. By contrast, sample functions for the Weight and Time dimensions exhibit non-linear behavior. These results are consistent with the differences in prediction performance between the non-linear function models and the linear approximations, as outlined in Table 1.

Several additional motion examples performed by dancers not included in the training data were used to evaluate the complete pipeline of the motion synthesis algorithm. The input sequences were always correctly segmented by the dynamic programming algorithm into the five fragments associated with the actions in the phrase. Kernel ridge regression was used to estimate the values of $\alpha^*, k^*, l^*$ as to minimize Equation 12 for different user-specified LMA-Effort vectors $\bar{\mathbf{e}}$. The recovered parameter values were used to synthesize animations with the specified desired styles. Videos of these automatically generated motions as well as additional results can be viewed at `http://movement.nyu.edu/learning-motion-styles/` . In order to test the generalization ability of our system, the target styles in this experiment were chosen to be considerably different from those in the training set. All of the synthesized sequences were visually inspected by LMA experts and, for the great majority, they were found to be consistent with the style target labels.

## 9   Discussions and Future Work

We have presented a novel technique that learns motion style synthesis from artificially-generated examples. Animations produced by our system have quality similar to pure motion capture play-back. Furthermore, we have shown that, even with a small database, it is possible to use pair-wise interpolation or extrapolation to generate new styles. In previous LMA-based animation systems [3], heuristic and hand-designed rules have been adopted to implement the style changes associated to LMA-Effort variations. To the best of our knowledge, our work represents the first attempt at automatically learning the mapping between LMA attributes and animation parameters. Although

our algorithm has shown to produce good results with small training data, we expect that larger databases with a wider variety of motion contents and styles are needed in order to build an effective animation system. Multi-way, as opposed to pair-wise, interpolation might lead to synthesis of more varied motion styles. Our approach could be easily generalized to other languages and notations, and to additional domains, such as facial animation. Our future work will focus on the recognition of LMA categories in motion capture data. Research in this area might point to methods for learning person-specific styles and to techniques for transferring individual movement signatures to arbitrary motion sequences.

### Acknowledgments

This work was carried out while LT was at Stanford University and visiting New York University. Thanks to Alyssa Lees for her help on this project and paper. We are grateful to Edward Warburton, Kevin Feeley, and Robb Bifano for assistance with the experimental setup and to Jared Silver for the Maya animations. Special thanks to Jan Burkhardt, Begonia Caparros, Ed Groff, Ellen Goldman and Pamela Schick for LMA observations and notations. This work has been supported by the National Science Foundation.

## References

[1] O. Arikan and D. A. Forsyth. Synthesizing constrained motions from examples. *ACM Transactions on Graphics*, 21(3):483–490, July 2002.

[2] M. Brand and A. Hertzmann. Style machines. In *Proceedings of ACM SIGGRAPH 2000*, Computer Graphics Proceedings, Annual Conference Series, pages 183–192, July 2000.

[3] D. Chi, M. Costa, L. Zhao, and N. Badler. The emote model for effort and shape. In *Proceedings of ACM SIGGRAPH 2000*, Computer Graphics Proceedings, Annual Conference Series, July 2000.

[4] N. Cristianini and J. Shawe-Taylor. *An Introduction to Support Vector Machines (and other kernel-based learning methods)*. Cambridge University Press, 2000.

[5] H. Drucker, C. J. C. Burges, L. Kaufman, A. Smola, and V. Vapnik. Support vector regression machines. In *Proc. NIPS 9*, 2003.

[6] M. A. Giese and T. Poggio. Morphable models for the analysis and synthesis of complex motion patterns. *International Journal of Computer Vision*, 38(1):59–73, 2000.

[7] P. Hackney. *Making Connections: Total Body Integration Through Bartenieff Fundamentals*. Routledge, 2000.

[8] M. T. Heath. *Scientific Computing: An Introductory Survey, Second edition*. McGraw Hill, 2002.

[9] E. Hsu, K. Pulli, and J. Popovic. Style translation for human motion. *ACM Transactions on Graphics*, 24(3):1082–1089, 2005.

[10] L. Kovar and M. Gleicher. Automated extraction and parameterization of motions in large data sets. *ACM Transactions on Graphics*, 23(3):559–568, Aug. 2004.

[11] J. Lee, J. Chai, P. S. A. Reitsma, J. K. Hodgins, and N. S. Pollard. Interactive control of avatars animated with human motion data. *ACM Transactions on Graphics*, 21(3):491–500, July 2002.

[12] Y. Li, T. Wang, and H.-Y. Shum. Motion texture: A two-level statistical model for character motion synthesis. *ACM Transactions on Graphics*, 21(3):465–472, July 2002.

[13] R. Murray, Z. Li, and S. Sastry. *A Mathematical Introduction to Robotic Manipulation*. CRC Press, 1994.

[14] H. Ney. The use of a one–stage dynamic programming algorithm for connected word recognition. *IEEE Transactions on Acoustics, Speech, and Signal Processing*, 32(3):263–271, 1984.

[15] K. Pullen and C. Bregler. Motion capture assisted animation: Texturing and synthesis. *ACM Transactions on Graphics*, 21(3):501–508, July 2002.

[16] C. Rose, M. Cohen, and B. Bodenheimer. Verbs and adverbs: multidimensional motion interpolation. *IEEE Computer Graphics and Application*, 18(5):32–40, 1998.

[17] V. N. Vapnik. *The Nature of Statistical Learning Theory*. Springer, 1995.

[18] D. J. Wiley and J. K. Hahn. Interpolation synthesis of articulated figure motion. *IEEE Computer Graphics and Application*, 17(6):39–45, 1997.
